# An Application of Boosting to Graph Classification

**Taku Kudo,     Eisaku Maeda**
NTT Communication Science Laboratories.
2-4 Hikaridai, Seika-cho, Soraku, Kyoto, Japan
{taku,maeda}@cslab.kecl.ntt.co.jp

**Yuji Matsumoto**
Nara Institute of Science and Technology.
8916-5 Takayama-cho, Ikoma, Nara, Japan
matsu@is.naist.jp

## Abstract

This paper presents an application of Boosting for classifying labeled graphs, general structures for modeling a number of real-world data, such as chemical compounds, natural language texts, and bio sequences. The proposal consists of i) decision stumps that use subgraph as features, and ii) a Boosting algorithm in which subgraph-based decision stumps are used as weak learners. We also discuss the relation between our algorithm and SVMs with convolution kernels. Two experiments using natural language data and chemical compounds show that our method achieves comparable or even better performance than SVMs with convolution kernels as well as improves the testing efficiency.

## 1   Introduction

Most machine learning (ML) algorithms assume that given instances are represented in numerical vectors. However, much real-world data is not represented as numerical vectors, but as more complicated structures, such as sequences, trees, or graphs. Examples include biological sequences (e.g., DNA and RNA), chemical compounds, natural language texts, and semi-structured data (e.g., XML and HTML documents).

Kernel methods, such as support vector machines (SVMs) [11], provide an elegant solution to handling such structured data. In this approach, instances are implicitly mapped into a high-dimensional space, where information about their similarities (inner-products) is only used for constructing a hyperplane for classification. Recently, a number of kernels have been proposed for such structured data, such as sequences [7], trees [2, 5], and graphs [6]. Most are based on the idea that a feature vector is implicitly composed of the counts of substructures (e.g., subsequences, subtrees, subpaths, or subgraphs).

Although kernel methods show remarkable performance, their implicit definitions of feature space make it difficult to know what kind of features (substructures) are relevant or which features are used in classifications. To use ML algorithms for data mining or as knowledge discovery tools, they must output a list of relevant features (substructures). This information may be useful not only for a detailed analysis of individual data but for the human decision-making process.

In this paper, we present a new machine learning algorithm for classifying labeled graphs that has the following characteristics: 1) It performs learning and classification using the

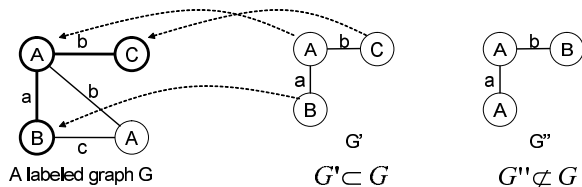

Figure 1: Labeled connected graphs and subgraph relation

structural information of a given graph. 2) It uses a set of all subgraphs (bag-of-subgraphs) as a feature set without any constraints, which is essentially the same idea as a convolution kernel [4]. 3) Even though the size of the candidate feature set becomes quite large, it *automatically* selects a compact and relevant feature set based on Boosting.

## 2 Classifier for Graphs

We first assume that an instance is represented in a labeled graph. The focused problem can be formalized as a general problem called the *graph classification problem*. The graph classification problem is to induce a mapping $f(\mathbf{x}) : \mathcal{X} \rightarrow \{\pm 1\}$, from given training examples $T = \{\langle \mathbf{x}_i, y_i \rangle\}_{i=1}^{L}$, where $\mathbf{x}_i \in \mathcal{X}$ is a labeled graph and $y_i \in \{\pm 1\}$ is a class label associated with the training data. We here focus on the problem of binary classification. The important characteristic is that input example $\mathbf{x}_i$ is represented not as a numerical feature vector but as a labeled graph.

### 2.1 Preliminaries

In this paper we focus on undirected, labeled, and connected graphs, since we can easily extend our algorithm to directed or unlabeled graphs with minor modifications. Let us introduce a labeled connected graph (or simply a labeled graph), its definitions and notations.

**Definition 1** *Labeled Connected Graph*
*A labeled graph is represented in a 4-tuple $G = (V, E, \mathcal{L}, l)$, where $V$ is a set of vertices, $E \subseteq V \times V$ is a set of edges, $\mathcal{L}$ is a set of labels, and $l : V \cup E \rightarrow \mathcal{L}$ is a mapping that assigns labels to the vertices and the edges. A labeled connected graph is a labeled graph such that there is a path between any pair of verticies.*

**Definition 2** *Subgraph*
*Let $G' = (V', E', \mathcal{L}', l')$ and $G = (V, E, \mathcal{L}, l)$ be labeled connected graphs. $G'$ matches $G$, or $G'$ is a subgraph of $G$ ($G' \subseteq G$) if the following conditions are satisfied: (1) $V' \subseteq V$, (2) $E' \subseteq E$, (3) $\mathcal{L}' \subseteq \mathcal{L}$, and (4) $l' = l$. If $G'$ is a subgraph of $G$, then $G$ is a supergraph of $G'$.*

Figure 1 shows an example of a labeled graph and its subgraph and non-subgraph.

### 2.2 Decision Stumps

Decision stumps are simple classifiers in which the final decision is made by a single hypothesis or feature. Boostexter [10] uses word-based decision stumps for text classification. To classify graphs, we define the subgraph-based decision stumps as follows.

**Definition 3** *Decision Stumps for Graphs*
*Let $t$ and $\mathbf{x}$ be labeled graphs and $y$ be a class label ($y \in \{\pm 1\}$). A decision stump classifier for graphs is given by*

$$h_{\langle t,y \rangle}(\mathbf{x}) \stackrel{\text{def}}{=} \begin{cases} y & t \subseteq \mathbf{x} \\ -y & otherwise. \end{cases}$$

The parameter for classification is a tuple $\langle t, y \rangle$, hereafter referred to as a *rule* of decision stumps. The decision stumps are trained to find a rule $\langle \hat{t}, \hat{y} \rangle$ that minimizes the error rate for the given training data $T = \{\langle \mathbf{x}_i, y_i \rangle\}_{i=1}^{L}$:

$$\langle \hat{t}, \hat{y} \rangle = \underset{t \in \mathcal{F}, y \in \{\pm 1\}}{\operatorname{argmin}} \frac{1}{L} \sum_{i=1}^{L} I(y_i \neq h_{\langle t, y \rangle}(\mathbf{x}_i)) = \underset{t \in \mathcal{F}, y \in \{\pm 1\}}{\operatorname{argmin}} \frac{1}{2L} \sum_{i=1}^{L} (1 - y_i h_{\langle t, y \rangle}(\mathbf{x}_i)), \quad (1)$$

where $\mathcal{F}$ is a set of candidate graphs or a *feature set* (i.e., $\mathcal{F} = \bigcup_{i=1}^{L} \{t | t \subseteq \mathbf{x}_i\}$) and $I(\cdot)$ is the indicator function. The gain function for a rule $\langle t, y \rangle$ is defined as

$$gain(\langle t, y \rangle) \overset{\text{def}}{=} \sum_{i=1}^{L} y_i h_{\langle t, y \rangle}(\mathbf{x}_i). \quad (2)$$

Using the gain, the search problem (1) becomes equivalent to the problem: $\langle \hat{t}, \hat{y} \rangle = \operatorname{argmax}_{t \in \mathcal{F}, y \in \{\pm 1\}} gain(\langle t, y \rangle)$. In this paper, we use gain instead of error rate for clarity.

### 2.3 Applying Boosting

The decision stump classifiers are too inaccurate to be applied to real applications, since the final decision relies on the existence of a single graph. However, accuracies can be *boosted* by the Boosting algorithm [3, 10]. Boosting repeatedly calls a given *weak learner* and finally produces a hypothesis $f$, which is a linear combination of $K$ hypotheses produced by the weak learners, i,e.: $f(\mathbf{x}) = sgn(\sum_{k=1}^{K} \alpha_k h_{\langle t_k, y_k \rangle}(\mathbf{x}))$. A weak learner is built at each iteration $k$ with different distributions or weights $\mathbf{d}^{(k)} = (d_i^{(k)}, \ldots, d_L^{(k)})$ on the training data, where $\sum_{i=1}^{L} d_i^{(k)} = 1, d_i^{(k)} \geq 0$. The weights are calculated to concentrate more on hard examples than easy examples. To use decision stumps as the weak learner of Boosting, we redefine the gain function (2) as:

$$gain(\langle t, y \rangle) \overset{\text{def}}{=} \sum_{i=1}^{L} y_i d_i h_{\langle t, y \rangle}(\mathbf{x}_i). \quad (3)$$

In this paper, we use the AdaBoost algorithm, the original and the best known algorithm among many variants of Boosting. However, it is trivial to fit our decision stumps to other boosting algorithms, such as Arc-GV [1] and Boosting with soft margins [8].

## 3 Efficient Computation

In this section, we introduce an efficient and practical algorithm to find the optimal rule $\langle \hat{t}, \hat{y} \rangle$ from given training data. This problem is formally defined as follows.

**Problem 1** *Find Optimal Rule*
*Let $T = \{\langle \mathbf{x}_1, y_1, d_1 \rangle, \ldots, \langle \mathbf{x}_L, y_L, d_L \rangle\}$ be training data where $\mathbf{x}_i$ is a labeled graph, $y_i \in \{\pm 1\}$ is a class label associated with $\mathbf{x}_i$ and $d_i$ ($\sum_{i=1}^{L} d_i = 1, d_i \geq 0$) is a normalized weight assigned to $\mathbf{x}_i$. Given $T$, find the optimal rule $\langle \hat{t}, \hat{y} \rangle$ that maximizes the gain, i.e., $\langle \hat{t}, \hat{y} \rangle = \operatorname{argmax}_{t \in \mathcal{F}, y \in \{\pm 1\}} d_i y_i h_{\langle t, y \rangle}$, where $\mathcal{F} = \bigcup_{i=1}^{L} \{t | t \subseteq \mathbf{x}_i\}$.*

The most naive and exhaustive method in which we first enumerate *all* subgraphs $\mathcal{F}$ and then calculate the gains for all subgraphs is usually impractical, since the number of subgraphs is exponential to its size. We thus adopt an alternative strategy to avoid such exhaustive enumerations. The method to find the optimal rule is modeled as a variant of branch-and-bound algorithm and will be summarized as the following strategies: 1) Define

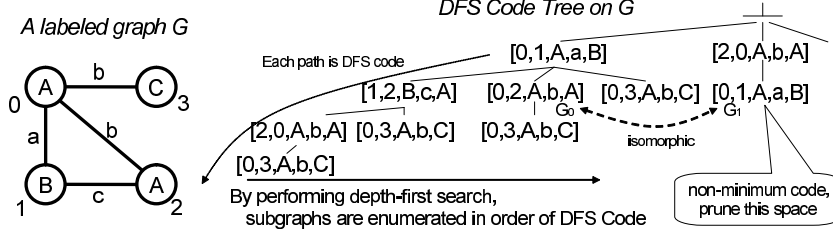

Figure 2: Example of DFS Code Tree for a graph

a canonical search space in which a whole set of subgraphs can be enumerated. 2) Find the optimal rule by traversing this search space. 3) Prune the search space by proposing a criteria for the upper bound of the *gain*. We will describe these steps more precisely in the next subsections.

## 3.1 Efficient Enumeration of Graphs

Yan et al. proposed an efficient depth-first search algorithm to enumerate all subgraphs from a given graph [12]. The key idea of their algorithm is a *DFS (depth first search) code*, a lexicographic order to the sequence of edges. The search tree given by the DFS code is called a *DFS Code Tree*. Leaving the details to [12], the order of the DFS code is defined by the lexicographic order of labels as well as the topology of graphs. Figure 2 illustrates an example of a DFS Code Tree. Each node in this tree is represented in a 5-tuple $[i, j, v_i, e_{ij}, v_j]$, where $e_{ij}$, $v_i$ and $v_j$ are the labels of $i$–$j$ edge, $i$-th vertex, and $j$-th vertex respectively. By performing a pre-order search of the DFS Code Tree, we can obtain all the subgraphs of a graph in order of their DFS code. However, one cannot avoid isomorphic enumerations even giving pre-order traverse, since one graph can have several DFS codes in a DFS Code Tree. So, canonical DFS code (minimum DFS code) is defined as its first code in the pre-order search of the DFS Code Tree. Yan et al. show that two graphs $G$ and $G'$ are isomorphic if and only if minimum DFS codes for the two graphs $min(G)$ and $min(G')$ are the same. We can thus ignore non-minimum DFS codes in subgraph enumerations. In other words, in depth-first traverse, we can prune a node with DFS code $c$, if $c$ is not minimum. The isomorphic graph represented in minimum code has already been enumerated in the depth-first traverse. For example, in Figure 2, if $G_1$ is identical to $G_0$, $G_0$ has been discovered before the node for $G_1$ is reached. This property allows us to avoid an explicit isomorphic test of the two graphs.

## 3.2 Upper bound of gain

DFS Code Tree defines a canonical search space in which one can enumerate all subgraphs from a given set of graphs. We consider an upper bound of the gain that allows pruning of subspace in this canonical search space. The following lemma gives a convenient method of computing a tight upper bound on $gain(\langle t', y \rangle)$ for any supergraph $t'$ of $t$.

**Lemma 1** *Upper bound of the gain:* $\mu(t)$
*For any* $t' \supseteq t$ *and* $y \in \{\pm 1\}$, *the gain of* $\langle t', y \rangle$ *is bounded by* $\mu(t)$ *(i.e.,* $gain(\langle t'y \rangle) \leq \mu(t)$*), where* $\mu(t)$ *is given by*

$$\mu(t) \overset{\text{def}}{=} \max\left( 2 \sum_{\{i|y_i=+1, t \subseteq \mathbf{x}_i\}} d_i - \sum_{i=1}^{L} y_i \cdot d_i, \ 2 \sum_{\{i|y_i=-1, t \subseteq \mathbf{x}_i\}} d_i + \sum_{i=1}^{L} y_i \cdot d_i \right).$$

**Proof 1**

$$gain(\langle t', y \rangle) = \sum_{i=1}^{L} d_i y_i h_{\langle t', y \rangle}(\mathbf{x}_i) = \sum_{i=1}^{L} d_i y_i \cdot y \cdot (2I(t' \subseteq \mathbf{x}_i) - 1),$$

*where $I(\cdot)$ is the indicator function. If we focus on the case $y = +1$, then*

$$
\begin{aligned}
gain(\langle t', +1 \rangle) &= 2 \sum_{\{i | t' \subseteq \mathbf{x}_i\}} y_i d_i - \sum_{i=1}^{L} y_i \cdot d_i \leq 2 \sum_{\{i | y_i = +1, t' \subseteq \mathbf{x}_i\}} d_i - \sum_{i=1}^{L} y_i \cdot d_i \\
&\leq 2 \sum_{\{i | y_i = +1, t \subseteq \mathbf{x}_i\}} d_i - \sum_{i=1}^{L} y_i \cdot d_i,
\end{aligned}
$$

*since $|\{i | y_i = +1, t' \subseteq \mathbf{x}_i\}| \leq |\{i | y_i = +1, t \subseteq \mathbf{x}_i\}|$   for any $t' \supseteq t$. Similarly,*

$$
gain(\langle t', -1 \rangle) \leq 2 \sum_{\{i | y_i = -1, t \subseteq \mathbf{x}_i\}} d_i + \sum_{i=1}^{L} y_i \cdot d_i.
$$

*Thus, for any $t' \supseteq t$ and $y \in \{\pm 1\}$, $gain(\langle t', y \rangle) \leq \mu(t)$.* □

We can efficiently prune the DFS Code Tree using the upper bound of gain $u(t)$. During pre-order traverse in a DFS Code Tree, we always maintain the temporally suboptimal gain $\tau$ among all the gains calculated previously. If $\mu(t) < \tau$, the gain of any supergraph $t' \supseteq t$ is no greater than $\tau$, and therefore we can safely prune the search space spanned from the subgraph $t$. If $\mu(t) \geq \tau$, then we cannot prune this space since a supergraph $t' \supseteq t$ might exist such that $gain(t') \geq \tau$.

### 3.3 Efficient Computation in Boosting

At each Boosting iteration, the suboptimal value $\tau$ is reset to 0. However, if we can calculate a tighter upper bound in advance, the search space can be pruned more effectively. For this purpose, a cache is used to maintain all rules found in the previous iterations. Suboptimal value $\tau$ is calculated by selecting one rule from the cache that maximizes the gain of the current distribution. This idea is based on our observation that a rule in the cache tends to be reused as the number of Boosting iterations increases. Furthermore, we also maintain the search space built by a DFS Code Tree as long as memory allows. This cache reduces duplicated constructions of a DFS Code Tree at each Boosting iteration.

## 4   Connection to Convolution Kernel

Recent studies [1, 9, 8] have shown that both Boosting and SVMs [11] work according to similar strategies: constructing an optimal hypothesis that maximizes the *smallest margin* between positive and negative examples. The difference between the two algorithms is the metric of margin; the margin of Boosting is measured in $l_1$-norm, while that of SVMs is measured in $l_2$-norm. We describe how maximum margin properties are translated in the two algorithms.

AdaBoost and Arc-GV asymptotically solve the following linear program, [1, 9, 8],

$$
\max_{\mathbf{w} \in I\!R^J, \rho \in I\!R^+} \rho; \quad s.t. \quad y_i \sum_{j=1}^{J} w_j h_j(\mathbf{x}_i) \geq \rho, \ ||\mathbf{w}||_1 = 1 \tag{4}
$$

where $J$ is the number of hypotheses. Note that in the case of decision stumps for graphs, $J = |\{\pm 1\} \times \mathcal{F}| = 2|\mathcal{F}|$.

SVMs, on the other hand, solve the following quadratic optimization problem [11]: [1]

$$
\max_{\mathbf{w} \in I\!R^J, \rho \in I\!R^+} \rho; \quad s.t. \quad y_i \cdot (\mathbf{w} \cdot \Phi(\mathbf{x}_i)) \geq \rho, \ ||\mathbf{w}||_2 = 1. \tag{5}
$$

The function $\Phi(\mathbf{x})$ maps the original input example $\mathbf{x}$ into a $J$-dimensional feature vector (i.e., $\Phi(\mathbf{x}) \in I\!R^J$). The $l_2$-norm margin gives the separating hyperplane expressed by dot-products in feature space. The feature space in SVMs is thus expressed implicitly by using a Marcer kernel function, which is a generalized dot-product between two objects, (i.e., $K(\mathbf{x}_1, \mathbf{x}_2) = \Phi(\mathbf{x}_1) \cdot \Phi(\mathbf{x}_2)$).

The best known kernel for modeling structured data is a convolution kernel [4] (e.g., string kernel [7] and tree kernel [2, 5]), which argues that a feature vector is implicitly composed of the counts of substructures. [2] The implicit mapping defined by the convolution kernel is given as: $\Phi(\mathbf{x}) = (\#(t_1 \subseteq \mathbf{x}), \ldots, \#(t_{|\mathcal{F}|} \subseteq \mathbf{x}))$, where $t_j \in \mathcal{F}$ and $\#(u)$ is the cardinality of $u$. Noticing that a decision stump can be expressed as $h_{\langle t, y \rangle}(\mathbf{x}) = y \cdot (2I(t \subseteq \mathbf{x}) - 1)$, we see that the constraints or feature space of Boosting with substructure-based decision stumps are essentially the same as those of SVMs with the convolution kernel [3]. The critical difference is the definition of margin: Boosting uses $l_1$-norm, and SVMs use $l_2$-norm. The difference between them can be explained by *sparseness*.

It is well known that the solution or separating hyperplane of SVMs is expressed in a linear combination of training examples using coefficients $\lambda$, (i.e., $\mathbf{w} = \sum_{i=1}^{L} \lambda_i \Phi(\mathbf{x}_i)$) [11]. Maximizing $l_2$-norm margin gives a sparse solution in the *example space*, (i.e., most of $\lambda_i$ becomes 0). Examples having non-zero coefficients are called *support vectors* that form the final solution. Boosting, in contrast, performs the computation explicitly in feature space. The concept behind Boosting is that only a few hypotheses are needed to express the final solution. $l_1$-norm margin realizes such a property [8]. Boosting thus finds a sparse solution in the *feature space*. The accuracies of these two methods depend on the given training data. However, we argue that Boosting has the following *practical* advantages. First, sparse hypotheses allow the construction of an efficient classification algorithm. The complexity of SVMs with tree kernel is $O(l|n_1||n_2|)$, where $n_1$ and $n_2$ are trees, and $l$ is the number of support vectors, which is too heavy to be applied to real applications. Boosting, in contrast, performs faster since the complexity depends only on a small number of decision stumps. Second, sparse hypotheses are useful in practice as they provide "transparent" models with which we can analyze how the model performs or what kind of features are useful. It is difficult to give such analysis with kernel methods since they define feature space implicitly.

## 5   Experiments and Discussion

To evaluate our algorithm, we employed two experiments using two real-world data.

(1) Cellphone review classification (**REV**)
The goal of this task is to classify reviews for cellphones as positive or negative. 5,741 sentences were collected from an Web-BBS discussion about cellphones in which users were directed to submit positive reviews separately from negative reviews. Each sentence is represented in a word-based dependency tree using a Japanese dependency parser CaboCha[4].

(2) Toxicology prediction of chemical compounds (**PTC**)
The task is to classify chemical compounds by carcinogenicity. We used the PTC data set[5] consisting of 417 compounds with 4 types of test animals: male mouse (MM), female

Table 1: Classification F-scores of the REV and PTC tasks

|  |  | REV | PTC | | | |
|---|---|---|---|---|---|---|
|  |  |  | MM | FM | MR | FR |
| Boosting | BOL-based Decision Stumps | 76.6 | 47.0 | 52.9 | 42.7 | 26.9 |
|  | Subgraph-based Decision Stumps | 79.0 | 48.9 | 52.5 | 55.1 | 48.5 |
| SVMs | BOL Kernel | 77.2 | 40.9 | 39.9 | 43.9 | 21.8 |
|  | Tree/Graph Kernel | 79.4 | 42.3 | 34.1 | 53.2 | 25.9 |

mouse (FM), male rat (MR) and female rat (FR). Each compound is assigned one of the following labels: {EE,IS,E,CE,SE,P,NE,N}. We here assume that CE,SE, and P are "positive" and that NE and NN are "negative", which is exactly the same setting as [6]. We thus have four binary classifiers (MM/FM/MR/FR) in this data set.

We compared the performance of our Boosting algorithm and support vector machines with tree kernel [2, 5] (for REV) and graph kernel [6] (for PTC) according to their F-score in 5-fold cross validation.

Table 1 summarizes the best results of REV and PCT task, varying the hyperparameters of Boosting and SVMs (e.g., maximum iteration of Boosting, soft margin parameter of SVMs, and termination probability of random walks in graph kernel [6]). We also show the results with bag-of-label (BOL) features as a baseline. In most tasks and categories, ML algorithms with structural features outperform the baseline systems (BOL). These results support our first intuition that structural features are important for the classification of structured data, such as natural language texts and chemical compounds.

Comparing our Boosting algorithm with SVMs using tree kernel, no significant difference can be found the REV data set. However, in the PTC task, our method outperforms SVMs using graph kernel on the categories MM, FM, and FR at a statistically significant level. Furthermore, the number of active features (subgraphs) used in Boosting is much smaller than those of SVMs. With our methods, about 1800 and 50 features (subgraphs) are used in the REV and PTC tasks respectively, while the potential number of features is quite large. Even giving all subgraphs as feature candidates, Boosting selects a small and highly relevant subset of features.

Figure 3 show an example of extracted support features (subgraphs) in the REV and PTC task respectively. In the REV task, features reflecting the domain knowledge (cellphone reviews) are extracted: 1) "*want to use* "→ positive, 2) "*hard to use*"→ negative, 3) "*recharging time is short*" → positive, 4) "*recharging time is long*" → negative. These features are interesting because we cannot determine the correct label (positive/negative) only using such bag-of-label features as "charging," "short," or "long." In the PTC task, similar structures show different behavior. For instance, Trihalomethanes (TTHMs), well-known carcinogenic substances (e.g., chloroform, bromodichloromethane, and chlorodi-bromomethane), contain the common substructure H-C-Cl (Fig. 3(a)). However, TTHMs do not contain the similar but different structure H-C(C)-Cl (Fig. 3(b)). Such structural information is useful for analyzing how the system classifies the input data in a category and what kind of features are used in the classification. We cannot examine such analysis in kernel methods, since they define their feature space implicitly.

The reason why graph kernel shows poor performance on the PTC data set is that it cannot identify subtle difference between two graphs because it is based on a random walks in a graph. For example, kernel dot-product between the similar but different structures 3(c) and 3(d) becomes quite large, although they show different behavior. To classify chemical compounds by their functions, the system must be capable of capturing subtle differences among given graphs.

The testing speed of our Boosting algorithm is also much faster than SVMs with tree/graph

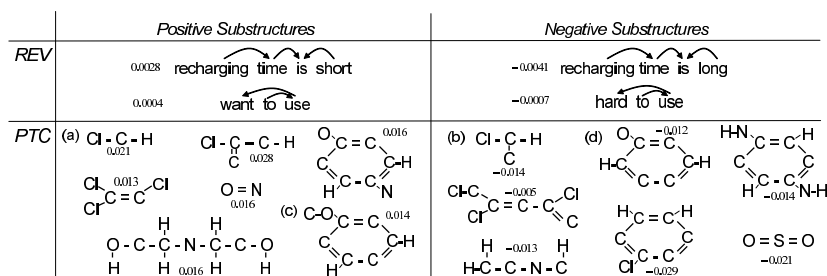

Figure 3: Support features and their weights

kernels. In the REV task, the speed of Boosting and SVMs are 0.135 sec./1,149 instances and 57.91 sec./1,149 instances respectively[6]. Our method is significantly faster than SVMs with tree/graph kernels without a discernible loss of accuracy.

## 6 Conclusions

In this paper, we focused on an algorithm for the classification of labeled graphs. The proposal consists of i) decision stumps that use subtrees as features, and ii) a Boosting algorithm in which subgraph-based decision stumps are applied as the weak learners. Two experiments are employed to confirm the importance of subgraph features. In addition, we experimentally show that our Boosting algorithm is accurate and efficient for classification tasks involving discrete structural features.

## Footnotes

[1]For simplicity, we omit the bias term ($b$) and the extension of Soft Margin.

[2] Strictly speaking, graph kernel [6] is not a convolution kernel because it is not based on the count of subgraphs, but on random walks in a graph.

[3] The difference between decision stumps and the convolution kernels is that the former uses a binary feature denoting the existence (or absence) of each substructure, whereas the latter uses the cardinality of each substructure. However, it makes little difference since a given graph is often sparse and the cardinality of substructures will be approximated by their existence.

[4] http://chasen.naist.jp/~taku/software/cabocha/

[5] http://www.predictive-toxicology.org/ptc/

[6]We tested the performances on Linux with XEON 2.4Ghz dual processors.

## References

[1] Leo Breiman. Prediction games and arching algoritms. *Neural Computation*, 11(7):1493 – 1518, 1999.

[2] Michael Collins and Nigel Duffy. Convolution kernels for natural language. In *NIPS 14, Vol.1*, pages 625–632, 2001.

[3] Yoav Freund and Robert E. Schapire. A decision-theoretic generalization of on-line learning and an application to boosting. *Journal of Computer and System Sicences*, 55(1):119–139, 1996.

[4] David Haussler. Convolution kernels on discrete structures. Technical report, UC Santa Cruz (UCS-CRL-99-10), 1999.

[5] Hisashi Kashima and Teruo Koyanagi. Svm kernels for semi-structured data. In *Proc. of ICML*, pages 291–298, 2002.

[6] Hisashi Kashima, Koji Tsuda, and Akihiro Inokuchi. Marginalized kernels between labeled graphs. In *Proc. of ICML*, pages 321–328, 2003.

[7] Huma Lodhi, Craig Saunders, John Shawe-Taylor, Nello Cristianini, and Chris Watkins. Text classification using string kernels. *Journal of Machine Learning Research*, 2, 2002.

[8] Gunnar. Rätsch, Takashi. Onoda, and Klaus-Robert Müller. Soft margins for AdaBoost. *Machine Learning*, 42(3):287–320, 2001.

[9] Robert E. Schapire, Yoav Freund, Peter Bartlett, and Wee Sun Lee. Boosting the margin: a new explanation for the effectiveness of voting methods. In *Proc. of ICML*, pages 322–330, 1997.

[10] Robert E. Schapire and Yoram Singer. BoosTexter: A boosting-based system for text categorization. *Machine Learning*, 39(2/3):135–168, 2000.

[11] Vladimir N. Vapnik. *Statistical Learning Theory*. Wiley-Interscience, 1998.

[12] Xifeng Yan and Jiawei Han. gspan: Graph-based substructure pattern mining. In *Proc. of ICDM*, pages 721–724, 2002.

